# Random Conic Pursuit for Semidefinite Programming

**Ariel Kleiner**
Computer Science Division
Univerisity of California
Berkeley, CA 94720
akleiner@cs.berkeley.edu

**Ali Rahimi**
Intel Research Berkeley
Berkeley, CA 94720
ali.rahimi@intel.com

**Michael I. Jordan**
Computer Science Division
University of California
Berkeley, CA 94720
jordan@cs.berkeley.edu

## Abstract

We present a novel algorithm, Random Conic Pursuit, that solves semidefinite programs (SDPs) via repeated optimization over randomly selected two-dimensional subcones of the PSD cone. This scheme is simple, easily implemented, applicable to very general SDPs, scalable, and theoretically interesting. Its advantages are realized at the expense of an ability to readily compute highly exact solutions, though useful approximate solutions are easily obtained. This property renders Random Conic Pursuit of particular interest for machine learning applications, in which the relevant SDPs are generally based upon random data and so exact minima are often not a priority. Indeed, we present empirical results to this effect for various SDPs encountered in machine learning; these experiments demonstrate the potential practical usefulness of Random Conic Pursuit. We also provide a preliminary analysis that yields insight into the theoretical properties and convergence of the algorithm.

## 1  Introduction

Many difficult problems have been shown to admit elegant and tractably computable representations via optimization over the set of positive semidefinite (PSD) matrices. As a result, semidefinite programs (SDPs) have appeared as the basis for many procedures in machine learning, such as sparse PCA [8], distance metric learning [24], nonlinear dimensionality reduction [23], multiple kernel learning [14], multitask learning [19], and matrix completion [2].

While SDPs can be solved in polynomial time, they remain computationally challenging. General-purpose solvers, often based on interior point methods, do exist and readily provide high-accuracy solutions. However, their memory requirements do not scale well with problem size, and they typically do not allow a fine-grained tradeoff between optimization accuracy and speed, which is often a desirable tradeoff in machine learning problems that are based on random data. Furthermore, SDPs in machine learning frequently arise as convex relaxations of problems that are originally computationally intractable, in which case even an exact solution to the SDP yields only an approximate solution to the original problem, and an approximate SDP solution can once again be quite useful. Although some SDPs do admit tailored solvers which are fast and scalable (e.g., [17, 3, 7]), deriving and implementing these methods is often challenging, and an easily usable solver that alleviates these issues has been elusive. This is partly the case because generic first-order methods do not apply readily to general SDPs.

In this work, we present Random Conic Pursuit, a randomized solver for general SDPs that is simple, easily implemented, scalable, and of inherent interest due to its novel construction. We consider general SDPs over $\mathbb{R}^{d \times d}$ of the form

$$\min_{X \succeq 0} \quad f(X) \quad \text{s.t.} \quad g_j(X) \leq 0, \qquad j = 1 \dots k, \tag{1}$$

where $f$ and the $g_j$ are convex real-valued functions, and $\succeq$ denotes the ordering induced by the PSD cone. Random Conic Pursuit minimizes the objective function iteratively, repeatedly randomly sampling a PSD matrix and optimizing over the random two-dimensional subcone given by this matrix and the current iterate. This construction maintains feasibility while avoiding the computational expense of deterministically finding feasible directions or of projecting into the feasible set. Furthermore, each iteration is computationally inexpensive, though in exchange we generally require a relatively large number of iterations. In this regard, Random Conic Pursuit is similar in spirit to algorithms such as online gradient descent and sequential minimal optimization [20] which have illustrated that in the machine learning setting, algorithms that take a large number of simple, inexpensive steps can be surprisingly successful.

The resulting algorithm, despite its simplicity and randomized nature, converges fairly quickly to useful approximate solutions. Unlike interior point methods, Random Conic Pursuit does not excel in producing highly exact solutions. However, it is more scalable and provides the ability to trade off computation for more approximate solutions. In what follows, we present our algorithm in full detail and demonstrate its empirical behavior and efficacy on various SDPs that arise in machine learning; we also provide early analytical results that yield insight into its behavior and convergence properties.

## 2 Random Conic Pursuit

Random Conic Pursuit (Algorithm 1) solves SDPs of the general form (1) via a sequence of simple two-variable optimizations (2). At each iteration, the algorithm considers the two-dimensional cone spanned by the current iterate, $X_t$, and a random rank one PSD matrix, $Y_t$. It selects as its next iterate, $X_{t+1}$, the point in this cone that minimizes the objective $f$ subject to the constraints $g_j(X_{t+1}) \leq 0$ in (1). The distribution of the random matrices is periodically updated based on the current iterate (e.g., to match the current iterate in expectation); these updates yield random matrices that are better matched to the optimum of the SDP at hand.

The two-variable optimization (2) can be solved quickly in general via a two-dimensional bisection search. As a further speedup, for many of the problems that we considered, the two-variable optimization can be altogether short-circuited with a simple check that determines whether the solution $X_{t+1} = X_t$, with $\hat{\beta} = 1$ and $\hat{\alpha} = 0$, is optimal. Additionally, SDPs with a trace constraint $\operatorname{tr} X = 1$ force $\alpha + \beta = 1$ and therefore require only a one-dimensional optimization.

Two simple guarantees for Random Conic Pursuit are immediate. First, its iterates are feasible for (1) because each iterate is a positive sum of two PSD matrices, and because the constraints $g_j$ of (2) are also those of (1). Second, the objective values decrease monotonically because $\beta = 1, \alpha = 0$ is a feasible solution to (2). We must also note two limitations of Random Conic Pursuit: it does not admit general equality constraints, and it requires a feasible starting point. Nonetheless, for many of the SDPs that appear in machine learning, feasible points are easy to identify, and equality constraints are either absent or fortuitously pose no difficulty.

We can gain further intuition by observing that Random Conic Pursuit's iterates, $X_t$, are positive weighted sums of random rank one matrices and so lie in the random polyhedral cones

$$\mathcal{F}_t^x := \left\{ \sum_{i=1}^{t} \gamma_i x_t x_t' : \gamma_i \geq 0 \right\} \subset \{X : X \succeq 0\}. \tag{3}$$

Thus, Random Conic Pursuit optimizes the SDP (1) by greedily optimizing $f$ w.r.t. the $g_j$ constraints within an expanding sequence of random cones $\{\mathcal{F}_t^x\}$. These cones yield successively better inner approximations of the PSD cone (a basis for which is the set of all rank one matrices) while allowing us to easily ensure that the iterates remain PSD.

In light of this discussion, one might consider approximating the original SDP by sampling a random cone $\mathcal{F}_n^x$ in one shot and replacing the constraint $X \succeq 0$ in (1) with the simpler linear constraints $X \in \mathcal{F}_n^x$. For sufficiently large $n$, $\mathcal{F}_n^x$ would approximate the PSD cone well (see Theorem 2 below), yielding an inner approximation that upper bounds the original SDP; the resulting problem would be easier than the original (e.g., it would become a linear program if the $g_j$ were linear). However, we have found empirically that a very large $n$ is required to obtain good approximations, thus negating any potential performance improvements (e.g., over interior point methods). Random Conic Pursuit

| **Algorithm 1:** | Random Conic Pursuit |
|---|---|
| | [brackets contain a particular, generally effective, sampling scheme] |

**Input**: A problem of the form (1)  $\quad n \in \mathbb{N}$: number of iterations
$\qquad X_0$: a feasible initial iterate $\quad [\kappa \in (0, 1)$: numerical stability parameter]
**Output**: An approximate solution $X_n$ to (1)

$p \leftarrow$ a distribution over $\mathbb{R}^d \quad [p \leftarrow \mathcal{N}(0, \Sigma)$ with $\Sigma = (1 - \kappa)X_0 + \kappa I_d]$
**for** $t \leftarrow 1$ **to** $n$ **do**
&emsp;Sample $x_t$ from $p$ and set $Y_t \leftarrow x_t x_t'$
&emsp;Set $\hat{\alpha}, \hat{\beta}$ to the optimizer of

$$\min_{\alpha, \beta \in \mathbb{R}} f(\alpha Y_t + \beta X_{t-1})$$
$$\text{s.t. } g_j(\alpha Y_t + \beta X_{t-1}) \leq 0, \quad j = 1 \ldots k \qquad (2)$$
$$\alpha, \beta \geq 0$$

&emsp;Set $X_t \leftarrow \hat{\alpha} Y_t + \hat{\beta} X_{t-1}$
&emsp;**if** $\hat{\alpha} > 0$ **then** Update $p$ based on $X_t \quad [p \leftarrow \mathcal{N}(0, \Sigma)$ with $\Sigma = (1 - \kappa)X_t + \kappa I_d]$
**end**
**return** $X_n$

successfully resolves this issue by iteratively expanding the random cone $\mathcal{F}_t^x$. As a result, we are able to much more efficiently access large values of $n$, though we compute a greedy solution within $\mathcal{F}_n^x$ rather than a global optimum over the entire cone. This tradeoff is ultimately quite advantageous.

## 3 Applications and Experiments

We assess the practical convergence and scaling properties of Random Conic Pursuit by applying it to three different machine learning tasks that rely on SDPs: distance metric learning, sparse PCA, and maximum variance unfolding. For each, we compare the performance of Random Conic Pursuit (implemented in MATLAB) to that of a standard and widely used interior point solver, `SeDuMi` [21] (via `cvx` [9]), and to the best available solver which has been customized for each problem.

To evaluate convergence, we first compute a ground-truth solution $X^*$ for each problem instance by running the interior point solver with extremely low tolerance. Then, for each algorithm, we plot the normalized objective value errors $[f(X_t) - f(X^*)]/|f(X^*)|$ of its iterates $X_t$ as a function of the amount of time required to generate each iterate. Additionally, for each problem, we plot the value of an application-specific metric for each iterate. These metrics provide a measure of the practical implications of obtaining SDP solutions which are suboptimal to varying degrees. We evaluate scaling with problem dimensionality by running the various solvers on problems of different dimensionalities and computing various metrics on the solver runs as described below for each experiment. Unless otherwise noted, we use the bracketed sampling scheme given in Algorithm 1 with $\kappa = 10^{-4}$ for all runs of Random Conic Pursuit.

### 3.1 Metric Learning

Given a set of datapoints in $\mathbb{R}^d$ and a pairwise similarity relation over them, metric learning extracts a Mahalanobis distance $d_A(x, y) = \sqrt{(x - y)'A(x - y)}$ under which similar points are nearby and dissimilar points are far apart [24]. Let $\mathcal{S}$ be the set of similar pairs of datapoints, and let $\bar{\mathcal{S}}$ be its complement. The metric learning SDP, for $A \in \mathbb{R}^{d \times d}$ and $C = \sum_{(i,j) \in \mathcal{S}} (x_i - x_j)(x_i - x_j)'$, is

$$\min_{A \succeq 0} \quad \text{tr}(CA) \quad \text{s.t.} \sum_{(i,j) \in \bar{\mathcal{S}}} d_A(x_i, x_j) \geq 1. \qquad (4)$$

To apply Random Conic Pursuit, $X_0$ is set to a feasible scaled identity matrix. We solve the two-variable optimization (2) via a double bisection search: at each iteration, $\alpha$ is optimized out with a one-variable bisection search over $\alpha$ given fixed $\beta$, yielding a function of $\beta$ only. This resulting function is itself then optimized using a bisection search over $\beta$.

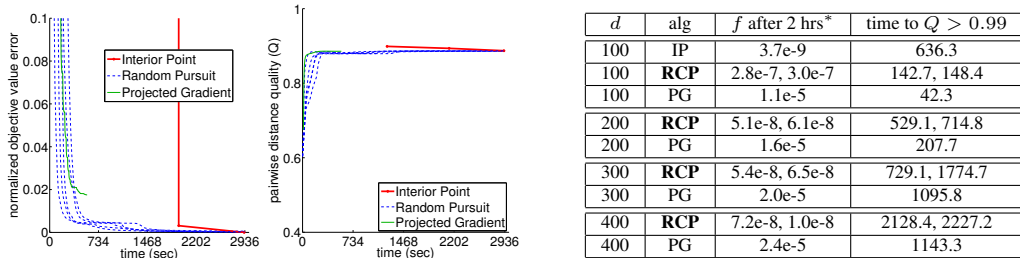

| $d$ | alg | $f$ after 2 hrs* | time to $Q > 0.99$ |
|---|---|---|---|
| 100 | IP | 3.7e-9 | 636.3 |
| 100 | **RCP** | 2.8e-7, 3.0e-7 | 142.7, 148.4 |
| 100 | PG | 1.1e-5 | 42.3 |
| 200 | **RCP** | 5.1e-8, 6.1e-8 | 529.1, 714.8 |
| 200 | PG | 1.6e-5 | 207.7 |
| 300 | **RCP** | 5.4e-8, 6.5e-8 | 729.1, 1774.7 |
| 300 | PG | 2.0e-5 | 1095.8 |
| 400 | **RCP** | 7.2e-8, 1.0e-8 | 2128.4, 2227.2 |
| 400 | PG | 2.4e-5 | 1143.3 |

Figure 1: Results for metric learning. **(plots)** Trajectories of objective value error (left) and $Q$ (right) on UCI ionosphere data. **(table)** Scaling experiments on synthetic data (IP = interior point, RCP = Random Conic Pursuit, PG = projected gradient), with two trials per $d$ for RCP and times in seconds. *For $d = 100$, third column shows $f$ after 20 minutes.

As the application-specific metric for this problem, we measure the extent to which the metric learning goal has been achieved: similar datapoints should be near each other, and dissimilar datapoints should be farther away. We adopt the following metric of quality of a solution matrix $X$, where $\zeta = \sum_i |\{j : (i,j) \in \mathcal{S}\}| \cdot |\{l : (i,l) \in \bar{\mathcal{S}}\}|$ and $1[\cdot]$ is the indicator function: $Q(X) = \frac{1}{\zeta} \sum_i \sum_{j:(i,j)\in\mathcal{S}} \sum_{l:(i,l)\in\bar{\mathcal{S}}} 1[d_{ij}(X) < d_{il}(X)]$.

To examine convergence behavior, we first apply the metric learning SDP to the UCI ionosphere dataset, which has $d = 34$ and 351 datapoints with two distinct labels ($\mathcal{S}$ contains pairs with identical labels). We selected this dataset from among those used in [24] because it is among the datasets which have the largest dimensionality and experience the greatest impact from metric learning in that work's clustering application. Because the interior point solver scales prohibitively badly in the number of datapoints, we subsampled the dataset to yield $4 \times 34 = 136$ datapoints.

To evaluate scaling, we use synthetic data in order to allow variation of $d$. To generate a $d$-dimensional data set, we first generate mixture centers by applying a random rotation to the elements of $\mathcal{C}_1 = \{(-1,1), (-1,-1)\}$ and $\mathcal{C}_2 = \{(1,1), (1,-1)\}$. We then sample each datapoint $x_i \in \mathbb{R}^d$ from $\mathcal{N}(0, I_d)$ and assign it uniformly at random to one of two clusters. Finally, we set the first two components of $x_i$ to a random element of $\mathcal{C}_k$ if $x_i$ was assigned to cluster $k \in \{1, 2\}$; these two components are perturbed by adding a sample from $\mathcal{N}(0, 0.25I_2)$.

The best known customized solver for the metric learning SDP is a projected gradient algorithm [24], for which we used code available from the author's website.

Figure 1 shows the results of our experiments. The two trajectory plots, for an ionosphere data problem instance, show that Random Conic Pursuit converges to a very high-quality solution (with high $Q$ and negligible objective value error) significantly faster than interior point. Additionally, our performance is comparable to that of the projected gradient method which has been customized for this task. The table in Figure 1 illustrates scaling for increasing $d$. Interior point scales badly in part because parsing the SDP becomes impracticably slow for $d$ significantly larger than 100. Nonetheless, Random Conic Pursuit scales well beyond that point, continuing to return solutions with high $Q$ in reasonable time. On this synthetic data, projected gradient appears to reach high $Q$ somewhat more quickly, though Random Conic Pursuit consistently yields significantly better objective values, indicating better-quality solutions.

## 3.2 Sparse PCA

Sparse PCA seeks to find a sparse unit length vector that maximizes $x'Ax$ for a given data covariance matrix $A$. This problem can be relaxed to the following SDP [8], for $X, A \in \mathbb{R}^{d \times d}$:

$$\min_{X \succeq 0} \quad \rho \mathbf{1}'|X|\mathbf{1} - \text{tr}(AX) \quad \text{s.t.} \quad tr(X) = 1, \tag{5}$$

where the scalar $\rho > 0$ controls the solution's sparsity. A subsequent rounding step returns the dominant eigenvector of the SDP's solution, yielding a sparse principal component.

We use the colon cancer dataset [1] that has been used frequently in past studies of sparse PCA and contains 2,000 microarray readings for 62 subjects. The goal is to identify a small number of

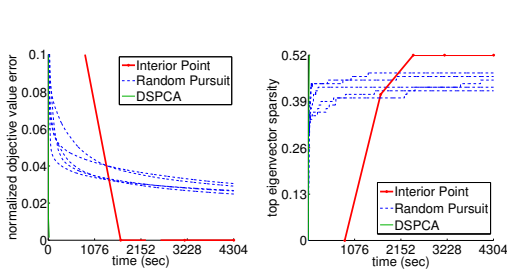

| $d$ | alg | $f$ after 4 hrs | sparsity after 4 hrs |
|---|---|---|---|
| 120 | IP | -10.25 | 0.55 |
| 120 | **RCP** | -9.98, -10.02 | 0.47, 0.45 |
| 120 | DSPCA | -10.24 | 0.55 |
| 200 | IP | failed | failed |
| 200 | **RCP** | -10.30, -10.27 | 0.51, 0.50 |
| 200 | DSPCA | -11.07 | 0.64 |
| 300 | IP | failed | failed |
| 300 | **RCP** | -9.39, -9.29 | 0.51, 0.51 |
| 300 | DSPCA | -11.52 | 0.69 |
| 500 | IP | failed | failed |
| 500 | **RCP** | -6.95, -6.54 | 0.53, 0.50 |
| 500 | DSPCA | -11.61 | 0.78 |

Figure 2: Results for sparse PCA. All solvers quickly yield similar captured variance (not shown here). **(plots)** Trajectories of objective value error (left) and sparsity (right), for a problem with $d = 100$. **(table)** Scaling experiments (IP = interior point, RCP = Random Conic Pursuit), with two trials per $d$ for RCP.

microarray cells that capture the greatest variance in the dataset. We vary $d$ by subsampling the readings and use $\rho = 0.2$ (large enough to yield sparse solutions) for all experiments.

To apply Random Conic Pursuit, we set $X_0 = A/\operatorname{tr}(A)$. The trace constraint (5) implies that $\operatorname{tr}(X_{t-1}) = 1$ and so $\operatorname{tr}(\alpha Y_t + \beta X_{t-1}) = \alpha \operatorname{tr}(Y_t) + \beta = 1$ in (2). Thus, we can simplify the two-variable optimization (2) to a one-variable optimization, which we solve by bisection search.

The fastest available customized solver for the sparse PCA SDP is an adaptation of Nesterov's smooth optimization procedure [8] (denoted by DSPCA), for which we used a MATLAB implementation with heavy MEX optimizations that is downloadable from the author's web site.

We compute two application-specific metrics which capture the two goals of sparse PCA: high captured variance and high sparsity. Given the top eigenvector $u$ of a solution matrix $X$, its captured variance is $u'Au$, and its sparsity is given by $\frac{1}{d}\sum_j 1[|u_j| < \tau]$; we take $\tau = 10^{-3}$ based on qualitative inspection of the raw microarray data covariance matrix $A$.

The results of our experiments are shown in Figure 2. As seen in the two plots, on a problem instance with $d = 100$, Random Conic Pursuit quickly achieves an objective value within 4% of optimal and thereafter continues to converge, albeit more slowly; we also quickly achieve fairly high sparsity (compared to that of the exact SDP optimum). In contrast, interior point is able to achieve lower objective value and even higher sparsity within the timeframe shown, but, unlike Random Conic Pursuit, it does not provide the option of spending less time to achieve a solution which is still relatively sparse. All of the solvers quickly achieve very similar captured variances, which are not shown. DSPCA is extremely efficient, requiring much less time than its counterparts to find nearly exact solutions. However, that procedure is highly customized (via several pages of derivation and an optimized implementation), whereas Random Conic Pursuit and interior point are general-purpose.

The table in Figure 2 illustrates scaling by reporting achieved objecive values and sparsities after the solvers have each run for 4 hours. Interior point fails due to memory requirements for $d > 130$, whereas Random Conic Pursuit continues to function and provide useful solutions, as seen from the achieved sparsity values, which are much larger than those of the raw data covariance matrix. Again, DSPCA continues to be extremely efficient.

### 3.3 Maximum Variance Unfolding (MVU)

MVU searches for a kernel matrix that embeds high-dimensional input data into a lower-dimensional manifold [23]. Given $m$ data points and a neighborhood relation $i \sim j$ between them, it forms their centered and normalized Gram matrix $G \in \mathbb{R}^{m \times m}$ and the squared Euclidean distances $d_{ij}^2 = G_{ii} + G_{jj} - 2G_{ij}$. The desired kernel matrix is the solution of the following SDP, where $X \in \mathbb{R}^{m \times m}$ and the scalar $\nu > 0$ controls the dimensionality of the resulting embedding:

$$\max_{X \succeq 0} \quad \operatorname{tr}(X) - \nu \sum_{i \sim j} (X_{ii} + X_{jj} - 2X_{ij} - d_{ij}^2)^2 \quad \text{s.t.} \quad \mathbf{1}'X\mathbf{1} = 0. \quad (6)$$

To apply Random Conic Pursuit, we set $X_0 = G$ and use the general sampling formulation in Algorithm 1 by setting $p = \mathcal{N}(0, \Pi(\nabla f(X_t)))$ in the initialization (i.e., $t = 0$) and update steps, where

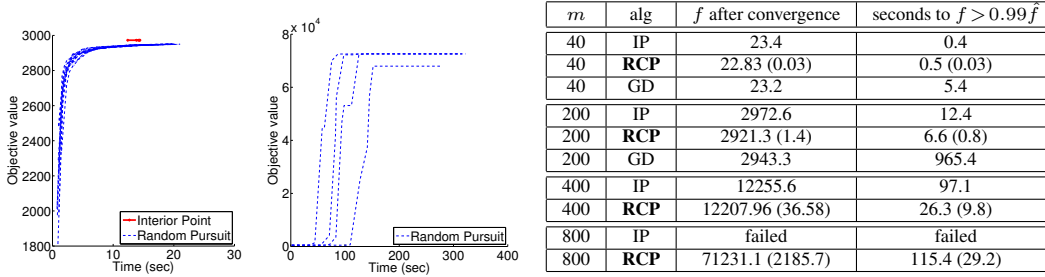

| $m$ | alg | $f$ after convergence | seconds to $f > 0.99\hat{f}$ |
|---|---|---|---|
| 40 | IP | 23.4 | 0.4 |
| 40 | **RCP** | 22.83 (0.03) | 0.5 (0.03) |
| 40 | GD | 23.2 | 5.4 |
| 200 | IP | 2972.6 | 12.4 |
| 200 | **RCP** | 2921.3 (1.4) | 6.6 (0.8) |
| 200 | GD | 2943.3 | 965.4 |
| 400 | IP | 12255.6 | 97.1 |
| 400 | **RCP** | 12207.96 (36.58) | 26.3 (9.8) |
| 800 | IP | failed | failed |
| 800 | **RCP** | 71231.1 (2185.7) | 115.4 (29.2) |

Figure 3: Results for MVU. **(plots)** Trajectories of objective value for $m = 200$ (left) and $m = 800$ (right). **(table)** Scaling experiments showing convergence as a function of $m$ (IP = interior point, RCP = Random Conic Pursuit, GD = gradient descent).

$\Pi$ truncates negative eigenvalues of its argument to zero. This scheme empirically yields improved performance for the MVU problem as compared to the bracketed sampling scheme in Algorithm 1. To handle the equality constraint, each $Y_t$ is first transformed to $\breve{Y}_t = (I - \mathbf{1}\mathbf{1}'/m)Y_t(I - \mathbf{1}\mathbf{1}'/m)$, which preserves PSDness and ensures feasibility. The two-variable optimization (2) proceeds as before on $\breve{Y}_t$ and becomes a two-variable quadratic program, which can be solved analytically.

MVU also admits a gradient descent algorithm, which serves as a straw-man large-scale solver for the MVU SDP. At each iteration, the step size is picked by a line search, and the spectrum of the iterate is truncated to maintain PSDness. We use $G$ as the initial iterate.

To generate data, we randomly sample $m$ points from the surface of a synthetic swiss roll [23]; we set $\nu = 1$. To quantify the amount of time it takes a solver to converge, we run it until its objective curve appears qualitatively flat and declare the convergence point to be the earliest iterate whose objective is within 1% of the best objective value seen so far (which we denote by $\hat{f}$).

Figure 3 illustrates that Random Conic Pursuit's objective values converge quickly, and on problems where the interior point solver achieves the optimum, Random Conic Pursuit nearly achieves that optimum. The interior point solver runs out of memory when $m > 400$ and also fails on smaller problems if its tolerance parameter is not tuned. Random Conic Pursuit easily runs on larger problems for which interior point fails, and for smaller problems its running time is within a small factor of that of the interior point solver; Random Conic Pursuit typically converges within 1000 iterations. The gradient descent solver is orders of magnitude slower than the other solvers and failed to converge to a meaningful solution for $m \geq 400$ even after 2000 iterations (which took 8 hours).

## 4 Analysis

Analysis of Random Conic Pursuit is complicated by the procedure's use of randomness and its handling of the constraints $g_j \leq 0$ explicitly in the sub-problem (2), rather than via penalty functions or projections. Nonetheless, we are able to obtain useful insights by first analyzing a simpler setting having only a PSD constraint. We thus obtain a bound on the rate at which the objective values of Random Conic Pursuit's iterates converge to the SDP's optimal value when the problem has no constraints of the form $g_j \leq 0$:

**Theorem 1** (Convergence rate of Random Conic Pursuit when $f$ is weakly convex and $k = 0$). *Let $f : \mathbb{R}^{d \times d} \to \mathbb{R}$ be a convex differentiable function with L-Lipschitz gradients such that the minimum of the following optimization problem is attained at some $X^*$:*

$$\min_{X \succeq 0} f(X). \tag{7}$$

*Let $X_1 \ldots X_t$ be the iterates of Algorithm 1 when applied to this problem starting at iterate $X_0$ (using the bracketed sampling scheme given in the algorithm specification), and suppose $\|X_t - X^*\|$ is bounded. Then*

$$\mathbb{E}f(X_t) - f(X^*) \leq \frac{1}{t} \cdot \max(\Gamma L, f(X_0) - f(X^*)), \tag{8}$$

*for some constant $\Gamma$ that does not depend on $t$.*

*Proof.* We prove that equation (8) holds in general for any $X^*$, and thus for the optimizer of $f$ in particular. The convexity of $f$ implies the following linear lower bound on $f(X)$ for any $X$ and $Y$:

$$f(X) \geq f(Y) + \langle \partial f(Y), X - Y \rangle. \tag{9}$$

The Lipschitz assumption on the gradient of $f$ implies the following quadratic upper bound on $f(X)$ for any $X$ and $Y$ [18]:

$$f(X) \leq f(Y) + \langle \partial f(Y), X - Y \rangle + \tfrac{L}{2}\|X - Y\|^2. \tag{10}$$

Define the random variable $\tilde{Y}_t := \gamma_t(Y_t)Y_t$ with $\gamma_t$ a positive function that ensures $\mathbb{E}\tilde{Y}_t = X^*$. It suffices to set $\gamma_t = q(Y)/\breve{p}(Y)$, where $\breve{p}$ is the distribution of $Y_t$ and $q$ is any distribution with mean $X^*$. In particular, the choice $\tilde{Y}_t := \gamma_t(x_t)x_t x_t'$ with $\gamma_t(x) = \mathcal{N}(x|0, X^*)/\mathcal{N}(x|0, \Sigma_t)$ satisfies this.

At iteration $t$, Algorithm 1 produces $\alpha_t$ and $\beta_t$ so that $X_{t+1} := \alpha_t Y_t + \beta_t X_t$ minimizes $f(X_{t+1})$. We will bound the defect $f(X_{t+1}) - f(X^*)$ at each iteration by sub-optimally picking $\hat{\alpha}_t = 1/t$, $\hat{\beta}_t = 1 - 1/t$, and $\hat{X}_{t+1} = \hat{\beta}_t X_t + \hat{\alpha}_t \gamma_t(Y_t)Y_t = \hat{\beta}_t X_t + \hat{\alpha}_t \tilde{Y}_t$. Conditioned on $X_t$, we have

$$\mathbb{E}f(X_{t+1}) - f(X^*) \leq \mathbb{E}f(\hat{\beta}_t X_t + \hat{\alpha}_t \tilde{Y}_t) - f(X^*) = \mathbb{E}f\left(X_t - \tfrac{1}{t}(X_t - \tilde{Y}_t)\right) - f(X^*) \tag{11}$$

$$\leq f(X_t) - f(X^*) + \mathbb{E}\left\langle \partial f(X_t), \tfrac{1}{t}(\tilde{Y}_t - X_t)\right\rangle + \tfrac{L}{2t^2}\mathbb{E}\|X_t - \tilde{Y}_t\|^2 \tag{12}$$

$$= f(X_t) - f(X^*) + \tfrac{1}{t}\langle \partial f(X_t), X^* - X_t \rangle + \tfrac{L}{2t^2}\mathbb{E}\|X_t - \tilde{Y}_t\|^2 \tag{13}$$

$$\leq f(X_t) - f(X^*) + \tfrac{1}{t}\left(f(X^*) - f(X_t)\right) + \tfrac{L}{2t^2}\mathbb{E}\|X_t - \tilde{Y}_t\|^2 \tag{14}$$

$$= \left(1 - \tfrac{1}{t}\right)\left(f(X_t) - f(X^*)\right) + \tfrac{L}{2t^2}\mathbb{E}\|X_t - \tilde{Y}_t\|^2. \tag{15}$$

The first inequality follows by the suboptimality of $\hat{\alpha}_t$ and $\hat{\beta}_t$, the second by Equation (10), and the third by (9).

Define $e_t := \mathbb{E}f(X_t) - f(X^*)$. The term $\mathbb{E}\|\tilde{Y}_t - X_t\|^2$ is bounded above by some absolute constant $\Gamma$ because $\mathbb{E}\|\tilde{Y}_t - X_t\|^2 = \mathbb{E}\|\tilde{Y}_t - X^*\|^2 + \|X_t - X^*\|^2$. The first term is bounded because it is the variance of $\tilde{Y}_t$, and the second term is bounded by assumption. Taking expectation over $X_t$ gives the bound $e_{t+1} \leq \left(1 - \tfrac{1}{t}\right)e_t + \tfrac{L\Gamma}{2t^2}$, which is solved by $e_t = \tfrac{1}{t} \cdot \max(\Gamma L, f(X_0) - f(X^*))$ [16]. □

Despite the extremely simple and randomized nature of Random Conic Pursuit, the theorem guarantees that its objective values converge at the rate $O(1/t)$ on an important subclass of SDPs. We omit here some readily available extensions: for example, the probability that a trajectory of iterates violates the above rate can be bounded by noting that the iterates' objective values behave as a finite difference sub-martingale. Additionally, the theorem and proof could be generalized to hold for a broader class of sampling schemes.

Directly characterizing the convergence of Random Conic Pursuit on problems with constraints appears to be significantly more difficult and seems to require introduction of new quantities depending on the constraint set (e.g., condition number of the constraint set and its overlap with the PSD cone) whose implications for the algorithm are difficult to explicitly characterize with respect to $d$ and the properties of the $g_j$, $X^*$, and the $Y_t$ sampling distribution. Indeed, it would be useful to better understand the limitations of Random Conic Pursuit. As noted above, the procedure cannot readily accommodate general equality constraints; furthermore, for some constraint sets, sampling only a rank one $Y_t$ at each iteration could conceivably cause the iterates to become trapped at a sub-optimal boundary point (this could be alleviated by sampling higher rank $Y_t$). A more general analysis is the subject of continuing work, though our experiments confirm empirically that we realize usefully fast convergence of Random Conic Pursuit even when it is applied to a variety of constrained SDPs.

We obtain a different analytical perspective by recalling that Random Conic Pursuit computes a solution within the random polyhedral cone $\mathcal{F}_n^x$, defined in (3) above. The distance between this cone and the optimal matrix $X^*$ is closely related to the quality of solutions produced by Random Conic Pursuit. The following theorem characterizes the distance between a sampled cone $\mathcal{F}_n^x$ and any fixed $X^*$ in the PSD cone:

**Theorem 2.** *Let $X^* \succ 0$ be a fixed positive definite matrix, and let $x_1, \ldots, x_n \in \mathbb{R}^d$ be drawn i.i.d. from $\mathcal{N}(0, \Sigma)$ with $\Sigma \succ X^*$. Then, for any $\delta > 0$, with probability at least $1 - \delta$,*

$$\min_{X \in \mathcal{F}_n^x} \|X - X^*\| \leq \frac{1 + \sqrt{2}\log\frac{1}{\delta}}{\sqrt{n}} \cdot \frac{2}{e}\sqrt{|\Sigma X^{*-1}|}\left\|\left(X^{*-1} - \Sigma^{-1}\right)^{-1}\right\|_2$$

See supplementary materials for proof. As expected, $\mathcal{F}_n^x$ provides a progressively better approximation to the PSD cone (with high probability) as $n$ grows. Furthermore, the rate at which this occurs depends on $X^*$ and its relationship to $\Sigma$; as the latter becomes better matched to the former, smaller values of $n$ are required to achieve an approximation of given quality.

The constant $\Gamma$ in Theorem 1 can hide a dependence on the dimensionality of the problem $d$, though the proof of Theorem 2 helps to elucidate the dependence of $\Gamma$ on $d$ and $X^*$ for the particular case when $\Sigma$ does not vary over time (the constants in Theorem 2 arise from bounding $\|\gamma_t(x_t)x_t x_t'\|$). A potential concern regarding both of the above theorems is the possibility of extremely adverse dependence of their constants on the dimensionality $d$ and the properties (e.g., condition number) of $X^*$. However, our empirical results in Section 3 show that Random Conic Pursuit does indeed decrease the objective function usefully quickly on real problems with relatively large $d$ and solution matrices $X^*$ which are rank one, a case predicted by the analysis to be among the most difficult.

## 5   Related Work

Random Conic Pursuit and the analyses above are related to a number of existing optimization and sampling algorithms.

Our procedure is closely related to feasible direction methods [22], which move along descent directions in the feasible set defined by the constraints at the current iterate. Cutting plane methods [11], when applied to some SDPs, solve a linear program obtained by replacing the PSD constraint with a polyhedral constraint. Random Conic Pursuit overcomes the difficulty of finding feasible descent directions or cutting planes, respectively, by sampling directions randomly and also allowing the current iterate to be rescaled.

Pursuit-based optimization methods [6, 13] return a solution within the convex hull of an *a priori*-specified convenient set of points $\mathcal{M}$. At each iteration, they refine their solution to a point between the current iterate and a point in $\mathcal{M}$. The main burden in these methods is to select a near-optimal point in $\mathcal{M}$ at each iteration. For SDPs having only a trace equality constraint and with $\mathcal{M}$ the set of rank one PSD matrices, Hazan [10] shows that such points in $\mathcal{M}$ can be found via an eigenvalue computation, thereby obtaining a convergence rate of $O(1/t)$. In contrast, our method selects steps randomly and still obtains a rate of $O(1/t)$ in the unconstrained case.

The Hit-and-Run algorithm for sampling from convex bodies can be combined with simulated annealing to solve SDPs [15]. In this configuration, similarly to Random Conic Pursuit, it conducts a search along random directions whose distribution is adapted over time.

Finally, whereas Random Conic Pursuit utilizes a randomized polyhedral inner approximation of the PSD cone, the work of Calafiore and Campi [5] yields a randomized outer approximation to the PSD cone obtained by replacing the PSD constraint $X \succeq 0$ with a set of sampled linear inequality constraints. It can be shown that for linear SDPs, the dual of the interior LP relaxation is identical to the exterior LP relaxation of the dual of the SDP. Empirically, however, this outer relaxation requires impractically many sampled constraints to ensure that the problem remains bounded and yields a good-quality solution.

## 6   Conclusion

We have presented Random Conic Pursuit, a simple, easily implemented randomized solver for general SDPs. Unlike interior point methods, our procedure does not excel at producing highly exact solutions. However, it is more scalable and provides useful approximate solutions fairly quickly, characteristics that are often desirable in machine learning applications. This fact is illustrated by our experiments on three different machine learning tasks based on SDPs; we have also provided a preliminary analysis yielding further insight into Random Conic Pursuit.

### Acknowledgments

We are grateful to Guillaume Obozinski for early discussions that motivated this line of work.

# References

[1] U. Alon, N. Barkai, D. A. Notterman, K. Gish, S. Ybarra, D. Mack, and A. J. Levine. Broad patterns of gene expression revealed by clustering analysis of tumor and normal colon tissues probed by oligonucleotide arrays. *Proc. Natl. Acad. Sci. USA*, 96:6745–6750, June 1999.

[2] S. Boyd and L. Vandenberghe. *Convex Optimization*. Cambridge University Press, 2004.

[3] S. Burer and R.D.C Monteiro. Local minima and convergence in low-rank semidefinite programming. *Mathematical Programming*, 103(3):427–444, 2005.

[4] S. Burer, R.D.C. Monteiro, and Y. Zhang. A computational study of a gradient-based log-barrier algorithm for a class of large-scale sdps. *Mathematical Programming*, 95(2):359–379, 2003.

[5] G. Calafiore and M.C. Campi. Uncertain convex programs: randomized solutions and confidence levels. *Mathematical Programming*, 102(1):25–46, 2005.

[6] K. Clarkson. Coresets, sparse greedy approximation, and the frank-wolfe algorithm. In *Symposium on Discrete Algorithms (SODA)*, 2008.

[7] A. d'Aspremont. Subsampling algorithms for semidefinite programming. Technical Report 0803.1990, ArXiv, 2009.

[8] A. d'Aspremont, L. El Ghaoui, M. I. Jordan, and G. R. G. Lanckriet. A direct formulation for sparse pca using semidefinite programming. *SIAM Review*, 49(3):434–448, 2007.

[9] M. Grant and S. Boyd. CVX: Matlab software for disciplined convex programming, version 1.21. http://cvxr.com/cvx, May 2010.

[10] E. Hazan. Sparse approximate solutions to semidefinite programs. In *Latin American conference on Theoretical informatics*, pages 306–316, 2008.

[11] C. Helmberg. A cutting plane algorithm for large scale semidefinite relaxations. In Martin Grötschel, editor, *The sharpest cut*, chapter 15. MPS/SIAM series on optimization, 2001.

[12] C. Helmberg and F. Rendl. A spectral bundle method for semidefinite programming. *SIAM Journal on Optimization archive*, 10(3):673–696, 1999.

[13] L. K. Jones. A simple lemma on greedy approximation in Hilbert space and convergence rates for projection pursuit regression and neural network training. *The Annals of Statistics*, 20(1):608–613, March 1992.

[14] G. R. G. Lanckriet, N. Cristianini, P. Bartlett, L. El Ghaoui, and M. I. Jordan. Learning the kernel matrix with semidefinite programming. *Journal of Machine Learning Research (JMLR)*, 5:27–72, December 2004.

[15] L. Lovász and S. Vempala. Fast algorithms for logconcave functions: Sampling, rounding, integration and optimization. In *Foundations of Computer Science (FOCS)*, 2006.

[16] A. Nemirovski, A. Juditsky, G. Lan, and A. Shapiro. Robust stochastic approximation approach to stochastic programming. *SIAM Journal on Optimization*, 19(4):1574–1609, 2009.

[17] Y Nesterov. Smooth minimization of non-smooth functions. *Mathematical Programming*, 103(1):127–152, May 2005.

[18] Y. Nesterov. Smoothing technique and its applications in semidefinite optimization. *Mathematical Programming*, 110(2):245–259, July 2007.

[19] G. Obozinski, B. Taskar, and M. I. Jordan. Joint covariate selection and joint subspace selection for multiple classification problems. *Statistics and Computing*, pages 1573–1375, 2009.

[20] J. Platt. Using sparseness and analytic QP to speed training of Support Vector Machines. In *Advances in Neural Information Processing Systems (NIPS)*, 1999.

[21] J.F. Sturm. Using sedumi 1.02, a matlab toolbox for optimization over symmetric cones. *Optimization Methods and Software, Special issue on Interior Point Methods*, 11-12:625–653, 1999.

[22] W. Sun and Y. Yuan. *Optimization Theory And Methods: Nonlinear Programming*. Springer Optimization And Its Applications, 2006.

[23] K. Q. Weinberger, F. Sha, Q. Zhu, and L. K. Saul. Graph laplacian regularization for large-scale semidefinite programming. In *Advances in Neural Information Processing Systems (NIPS)*, 2006.

[24] E. Xing, A. Ng, M. Jordan, and S. Russell. Distance metric learning, with application to clustering with side-information. In *Advances in Neural Information Processing Systems (NIPS)*, 2003.

